# Reducing Calibration Time For Brain-Computer Interfaces: A Clustering Approach

**Matthias Krauledat[1,2], Michael Schröder[2], Benjamin Blankertz[2], Klaus-Robert Müller[1,2]**

[1]Technical University Berlin, Str. des 17. Juni 135, 10 623 Berlin, Germany
[2] Fraunhofer FIRST.IDA, Kekuléstr. 7, 12 489 Berlin, Germany
`{kraulem,schroedm,blanker,klaus}@first.fhg.de`

## Abstract

Up to now even subjects that are experts in the use of machine learning based BCI systems still have to undergo a calibration session of about 20-30 min. From this data their (movement) intentions are so far infered. We now propose a new paradigm that allows to completely omit such calibration and instead transfer knowledge from prior sessions. To achieve this goal we first define normalized CSP features and distances in-between. Second, we derive prototypical features across sessions: (a) by clustering or (b) by feature concatenation methods. Finally, we construct a classifier based on these individualized prototypes and show that, indeed, classifiers can be successfully transferred to a new session for a number of subjects.

## 1 Introduction

BCI systems typically require training on the subject side and on the decoding side (e.g. [1, 2, 3, 4, 5, 6, 7]). While some approaches rely on operant conditioning with extensive subject training (e.g. [2, 1]), others, such as the Berlin Brain-Computer Interface (BBCI) put more emphasis on the machine side (e.g. [4, 8, 9]). But when following our philosophy of 'letting the machines learn', a calibration session of approximately 20-30 min was so far required, even for subjects that are beyond the status of BCI novices.

The present contribution studies to what extent we can *omit* this brief calibration period. In other words, is it possible to successfully transfer information from prior BCI sessions of the same subject that may have taken place days or even weeks ago? While this question is of high practical importance to the BCI field, it has so far only been addressed in [10] in the context of transfering channel selection results from subject to subject. In contrast to this prior approach, we will focus on the more general question of transfering whole classifiers, resp. individualized representations between sessions. Note that EEG (electroencephalogram) patterns typically vary strongly from one session to another, due to different psychological pre-conditions of the subject. A subject might for example show different states of fatigue and attention, or use diverse strategies for movement imagination across sessions. A successful session to session transfer should thus capture generic 'invariant' discriminative features of the BCI task.

For this we first transform the EEG feature set from each prior session into a 'standard' format (section 2) and normalize it. This allows to define a consistent measure that can quantify the distance between representations. We use CSP-based classifiers (see section 3.1 and e.g. [11]) for the discrimination of brain states; note that the line of thought presented here can also be pursued for other feature sets resp. for classifiers. Once a distance function (section 3.2) is established in CSP filter space, we can cluster existing CSP filters in order to obtain the most salient prototypical CSP-type filters for a subject across sessions (section 3.3). To this end, we use the IBICA algorithm [12, 13] for computing prototypes by a robust ICA decomposition (section 3.3). We will show that these new CSP prototypes are physiologically meaningful and furthermore are highly robust representations which are less easily distorted by noise artifacts.

## 2 Experiments and Data

Our BCI system uses Event-Related (De-)Synchronization (ERD/ERS) phenomena [3] in EEG signals related to hand and foot imagery as classes for control. The term refers to a de– or increasing band power in specific frequency bands of the EEG signal during the imagination of movements. These phenomena are well-studied and consistently reproducible features in EEG recordings, and are used as the basis of many BCI systems (e.g. [11, 14]). For the present study we investigate data from experiments with 6 healthy subjects: *aw* (13 sessions), *al* (8 sessions), *cm* (4 sessions), *ie* (4 sessions), *ay* (5 sessions) and *ch* (4 sessions). These are all the subjects that participated in at least 4 BCI sessions. Each session started with the recording of calibration data, followed by a machine learning phase and a feedback phase of varying duration. All following retrospective analyses were performed on the calibration data only.

During the experiments the subjects were seated in a comfortable chair with arm rests. For the recording of the calibration data every 4.5–6 seconds one of 3 different visual stimuli was presented, indicating a motor imagery task the subject should perform during the following 3–3.5 seconds. The randomized and balanced motor imagery tasks investigated for all subjects except *ay* were left hand (*l*), right hand (*r*), and right foot (*f*). Subject *ay* only performed left- and right hand tasks. Between 120 and 200 trials were performed during the calibration phase of one session for each motor imagery class.

Brain activity was recorded from the scalp with multi-channel EEG amplifiers using at least 64 channels. Besides EEG channels, we recorded the electromyogram (EMG) from both forearms and the right lower leg as well as horizontal and vertical electrooculogram (EOG) from the eyes. The EMG and EOG channels were exclusively used to ensure that the subjects performed no real limb or eye movements correlated with the mental tasks. As their activity can directly (via artifacts) or indirectly (via afferent signals from muscles and joint receptors) be reflected in the EEG channels they could be detected by the classifier. Controlling EMG and EOG ensured that the classifier operated on true EEG signals only.

### Data preprocessing and Classification

The time series data of each trial was windowed from 0.5 seconds after cue to 3 seconds after cue. The data of the remaining interval was band pass filtered between either 9 Hz – 25 Hz or 10 Hz – 25 Hz, depending on the signal characteristics of the subject. In any case the chosen spectral interval comprised the subject specific frequency bands that contained motor-related activity.

For each subject a subset of EEG channels was determined that had been recorded for all of the subject's sessions. These subsets typically contained 40 to 45 channels which were densely located (according to the international 10-20 system) over the more central areas of the scalp (see scalp maps in following sections). The EEG channels of each subject were reduced to the determined subset before proceeding with the calculation of Common Spatial Patterns (CSP) for different (subject specific) binary classification tasks.

After projection on the CSP filters, the bandpower was estimated by taking the logvariance over time. Finally, a linear discriminant analysis (LDA) classifier was applied to the best discriminable two-class combination.

## 3 A closer look at the CSP parameter space

### 3.1 Introduction of Common Spatial Patterns (CSP)

The common spatial pattern (CSP) algorithm is very useful in calculating spatial filters for detecting ERD/ERS effects ([15]) and can be applied to ERD-based BCIs, see [11]. It has been extended to multi-class problems in [14], and further extensions and robustifications concerning a simultaneous optimization of spatial and frequency filters were presented in [16, 17, 18]. Given two distributions in a high-dimensional space, the (supervised) CSP algorithm finds directions (i.e., spatial filters) that maximize variance for one class and simultaneously minimize variance for the other class. After having band-pass filtered the EEG signals to the rhythms of interest, high variance reflects a strong rhythm and low variance a weak (or attenuated) rhythm. Let us take the example of discriminating left hand vs. right hand imagery. The filtered signal corresponding to the desynchronization of the left hand motor cortex is characterized by a strong motor rhythm during imagination of right hand movements (left hand is in idle state), and by an attenuated motor rhythm during left hand imagination. This criterion is exactly what the CSP algorithm optimizes: maximizing variance for

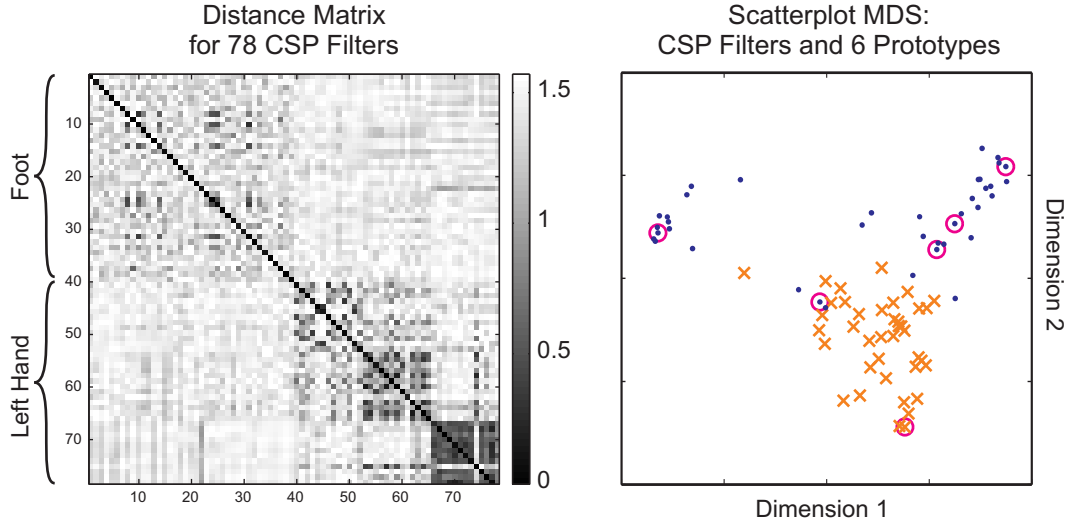

Figure 1: **Left:** Non-euclidean distance matrix for 78 CSP filters of imagined left hand and foot movement. **Right:** Scatterplot of the first vs. second dimension of CSP filters after Multi-Dimensional Scaling (MDS). Filters that minimize the variance for the imagined left hand are plotted as red crosses, foot movement imagery filters are shown as blue dots. Cluster centers detected by IBICA are marked with magenta circles. Both figures show data from *al*.

the class of right hand trials and at the same time minimizing variance for left hand trials. Furthermore the CSP algorithm calculates the dual filter that will focus on the area of the right hand and it will even calculate several filters for both optimizations by considering the remaining orthogonal subspaces.

Let $\Sigma_i$ be the covariance matrix of the trial-concatenated matrix of dimension [channels × concatenated time-points] belonging to the respective class $i \in \{1,2\}$. The CSP analysis consists of calculating a matrix $Q$ and diagonal matrix $D$ with elements in $[0,1]$ such that

$$Q\Sigma_1 Q^\top = D \qquad \text{and} \qquad Q\Sigma_2 Q^\top = I - D. \tag{1}$$

This can be solved as a generalized eigenvalue problem. The projection that is given by the *i*-th row of matrix $Q$ has a relative variance of $d_i$ (*i*-th element of $D$) for trials of class 1 and relative variance $1 - d_i$ for trials of class 2. If $d_i$ is near 1 the filter given by the *i*-th row of $Q$ maximizes variance for class 1, and since $1 - d_i$ is near 0, minimizes variance for class 2. Typically one would retain projections corresponding to the three highest eigenvalues $d_i$, i.e., CSP filters for class 1, and projections corresponding to the three lowest eigenvalues, i.e., CSP filters for class 2.

### 3.2 Comparison of CSP filters

Since the results of the CSP algorithm are the solutions of a generalized eigenvalue problem, where every multiple of an eigenvector is again a solution to the eigenvalue problem. If we want to compare different CSP filters, we must therefore keep in mind that every point on the line through a CSP filter point and the origin can be identified (except for the origin itself). More precisely, it is sufficient to consider only normalized CSP vectors on the (#channels-1)-dimensional hypersphere. This suggests that the CSP space is inherently non-euclidean. As a more appropriate metric between two points $c_1$ and $c_2$ in this space, we calculated the angle between the two lines corresponding to these points.

$$m(c_1, c_2) = \arccos(\frac{c_1 * c_2}{|c_1| * |c_2|})$$

When applying this measure to a set of CSP filters $(c_i)_{i \leq n}$, one can generate the distance matrix

$$D = (m(c_i, c_j))_{i,j \leq n},$$

which can then be used to find prototypical examples of CSP filters. Fig.1 shows an example of a distance matrix for 78 CSP filters for the discrimination of the variance during imagined left hand movement and foot movement. Based on the left hand signals, three CSP filters showing the lowest

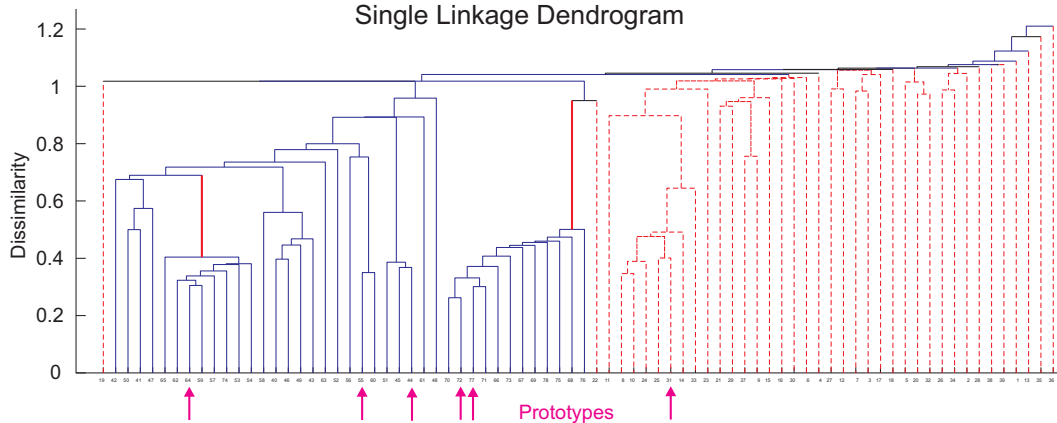

Figure 2: Dendrogram of a hierarchical cluster tree for the CSP filters of left hand movement imagery (dashed red lines) and foot movement imagery (solid blue lines). Cluster centers detected by IBICA are used as CSP prototypes. They are marked with magenta arrows.

eigenvalues were chosen for each of the 13 sessions. The same number of $3 \times 13$ filters were chosen for the foot signals. The filters are arranged in groups according to their relative magnitude of the eigenvalues, i.e., filters with the largest eigenvalues are grouped together, then filters with the second largest eigenvalues etc.

The distance matrix in Fig.1 shows a block structure which reveals that the filters of each group have low distances amongst each other as compared to the distances to members of other groups. This is especially true for filters for the minimization of variance in left hand trials.

### 3.3 Finding Clusters in CSP space

The idea to find CSP filters that recur in the processing of different sessions of a single subject is very appealing, since these filters can be re-used for efficient classification of unseen data. As an example of clustered parameters, Fig.2 shows a hierarchical clustering tree (see [19]) of CSP filters of different sessions for subject *al*. Single branches of the tree form distinct clusters, which are also clearly visible in a projection of the first Multi-Dimensional Scaling-Components in Fig.1 (for MDS, see [20]).

The proposed metric of section 3.2 coincides with the metric used for Inlier-Based Independent Component Analysis (IBICA, see [12, 13]). This method was originally intended to find estimators of the super-Gaussian source signals from a mixture of signals. By projecting the data onto the hypersphere and using the angle distance, it has been demonstrated that the correct source signals can be found even in high-dimensional data. The key ingredient of this method is the robust identification of inlier points as it can be done with the $\gamma$-index (see [21]), which is defined as follows:

Let $z \in \{c_1, \dots, c_n\}$ be a point in CSP-space, and let $nn_1(z), \dots, nn_k(z)$ be the $k$ nearest neighbors of $z$, according to the distance $m$. We then call the average distance of $z$ to its neighbors the $\gamma$-index of $z$, i.e.

$$\gamma(z) = \frac{1}{k} \sum_{j=1}^{k} m(z, nn_j(z)).$$

If $z$ lies in a densely populated region of the hypersphere, then the average distance to its neighbors is small, whereas if it lies in a sparse region, the average distance is high. The data points with the smallest $\gamma$ are good candidates for prototypical CSP filters since they are similar to other filters in the comparison set. This suggests that these filters are good solutions in a number of experiments and are therefore robust against changes in the data such as outliers, variations in background noise etc.

## 4 Competing analysis methods: How much training is needed?

Fig.3 shows an overview of the validation methods used for the algorithms under study. The left part shows validation methods which mimick the following BCI scenario: a new session starts and no

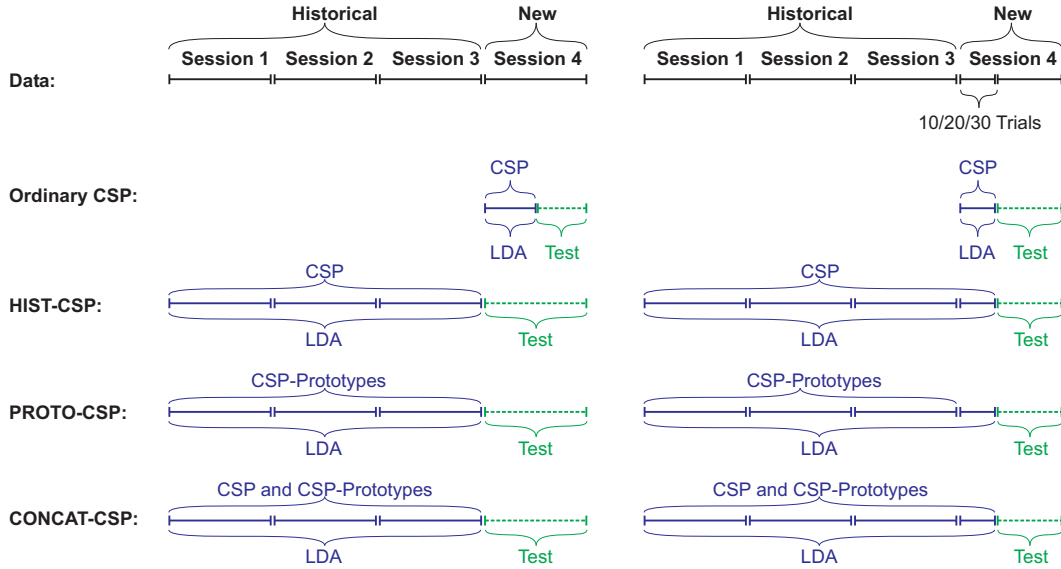

Figure 3: Overview of the presented training and testing modes for the example of four available sessions. The left part shows a comparison of ordinary CSP with three methods that do not require calibration. The validation scheme in the right part compares CSP with three adaptive methods. See text for details.

data has been collected yet. The top row represents data of all sessions in original order. Later rows describe different data splits for the training of the CSP filters and LDA (both depicted in blue solid lines) and for the testing of the trained algorithms on unseen data (green dashed lines). The ordinary CSP method does not take any historical data from prior sessions into account (second row). It uses training data only from the first half of the current session. This serves as a baseline to show the general quality of the data, since half of the session data is generally enough to train a classifier that is well adapted to the second half of the session. Note that this evaluation only corresponds to a real BCI scenario where many calibration trials of the same day are available.

## 4.1 Zero training methods

This is contrasted to the following rows, which show the exclusive use of historic data in order to calculate LDA and one single set of CSP filters from the collected data of all prior sessions (third row), or calculate one set of CSP filters for each historic session and derive prototypical filters from this collection as described in section 3.3 (fourth row), or use a combination of row three and four that results in a concatenation of CSP filters and derived CSP prototypes (fifth row). Feature concatenation is an effective method that has been shown to improve CSP-based classifiers considerably (see [22]).

## 4.2 Adaptive training methods

The right part of Fig.3 expands the training sets for rows three, four and five for the first 10, 20 or 30 trials per class of the data of the new session. In the methods of row 4 and 5, only LDA profits from the new data, whereas CSP prototypes are calculated exclusively on historic data as before. This approach is compared against the ordinary CSP approach that now only uses the same small amount of training data from the new session.

This scheme, as well as the one presented in section 4.1, has been cross-validated such that each available session was used as a test session instead of the last one.

## 5 Results

The underlying question of this paper is whether information gathered from previous experimental sessions can prove its value in a new session. In an ideal case existing CSP filters and LDA classifiers could be used to start the feedback phase of the new session immediately, without the need to collect new calibration data.

| Subjects | aw | al | cm | ie | ay | ch |
|---|---|---|---|---|---|---|
| Classes | LF | RF | LF | LR | LR | LR |
| Ordinary CSP | 5.0 | 2.7 | 11.8 | 16.2 | 11.7 | 6.2 |
| HIST | 10.1 | 2.9 | 23.0 | 26.0 | 13.3 | **6.9** |
| PROTO | 9.9 | 3.1 | 21.5 | 26.2 | **10.0** | 11.4 |
| CONCAT | **8.9** | **2.7** | **19.5** | **23.7** | 12.4 | 7.4 |
| Sessions | 13 | 7 | 4 | 4 | 5 | 4 |

Table 1: Results of Zero-Training modes. All classification errors are given in %. While the ordinary CSP method uses half of the new session for training, the three methods HIST, PROTO and CONCAT exclusively use historic data for the calculation of CSP filters and LDA. (as described on the left side of Fig.3). Amongst them, CONCAT performs best in four of the six subjects. For subjects *al*, *ay* and *ch* its result is even comparable to that of ordinary CSP.

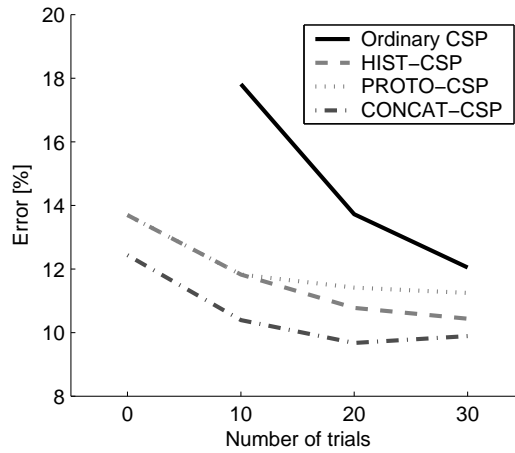

Figure 4: Incorporating more and more data from the current session (10, 20 or 30 trials per class), the classification error decreases for all of the four methods described on the right side of Fig.3. The three methods HIST, PROTO and CONCAT clearly outperform ordinary CSP. Interestingly the best zero-training method CONCAT is only outperformed by ordinary CSP if the latter has a head start of 30 trials per class.

We checked for the validity of this scenario based on the data described in section 2. Table 1 shows the classification results for the different classification methods under the Zero-training validation scheme. For subjects *al*, *ay* and *ch*, the classification error of CONCAT is of the same magnitude as the ordinary (training-based) CSP-approach. For the other three subjects, CONCAT outperforms the methods HISTand PROTO. Although the ideal case is not reached for every subject, the table shows that our proposed methods provide a decent step towards the goal of Zero-training for BCI.

Another way to at least reduce the necessary preparation time for a new experimental session is to record only very few new trials and combine them with data from previous sessions in order to get a quicker start. We simulate this strategy by allowing the new methods HIST, PROTO and CONCAT to take a look also on the first 10, 20 or 30 trials per class of the new session. The baseline to compare their performance would be a BCI system trained only on these initial trials. In Fig. 4, this comparison is depicted. Here the influence of the number of initial training trials becomes visible. If no new data is available, the ordinary classification approach of course can not produce any output, whereas the history-based methods, e.g. CONCAT already generates a stable estimation of the class labels. All methods gain performance in terms of smaller test errors as more and more trials are added. Only after training on at least 30 trials per class, ordinary CSP reaches the classification level that CONCAT had already shown without any training data of the current session.

Fig.5 shows some prototypical CSP filters as detected by IBICA clustering for subject *al* and left hand vs. foot motor imagery. All filters have small support (i.e., many entries are close to 0), and the few large entries are located on neurophysiologically important areas: Filters 1–2 and 4–6 cover the motor cortices corresponding to imagined hand movements, while filter 3 focuses on the central foot area. This shows that the cluster centers are spatial filters that meet our neurophysiological ex-

CSP Prototype Filters

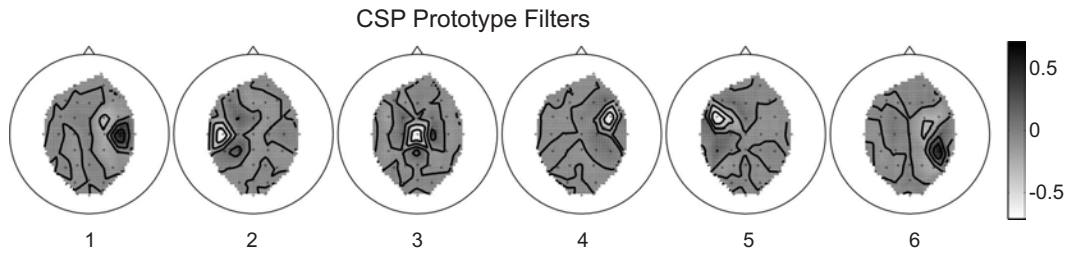

Figure 5: First six CSP prototype filters determined by IBICA for *al*.

pectations, since they are able to capture the frequency power modulations over relevant electrodes, while masking out unimportant or noisy channels.

## 6 Discussion and Conclusion

Advanced BCI systems (e.g. BBCI) recently aquired the ability to dispense with extensive subject training and now allow to infer a blueprint of the subject's volition from a short calibration session of approximately 30 min. This became possible through the use of modern machine learning technology. The next step along this line to make BCI more practical is to strive for zero calibration time. Certainly it will not be realistic to achieve this goal for arbitrary BCI novices, rather in this study we have concentrated on experienced BCI users (with 4 and more sessions) and discussed algorithms to re-use their classifiers from prior sessions. Note that the construction of a classifier that is *invariant* against session to session changes, say, due to different vigilance, focus or motor imagination across sessions is a hard task.

Our contribution shows that experienced BCI subjects do not necessarily need to perform a new calibration period in a new experiment. By analyzing the CSP parameter space, we could reveal an appropriate characterization of CSP filters. Finding clusters of CSP parameters for old sessions, novel prototypical CSP filters can be derived, for which the neurophysiological validity could be shown exemplarily. The concatenation of these prototype filters with some CSP filters trained on the same amount of data results in a classifier that not only performs comparable to the presented ordinary CSP approach (trained on a large amount of data from the same session) in half of the subjects, but also outperforms ordinary CSP considerably when only few data points are at hand. This means that experienced subjects are predictable to an extent that they do not require calibration anymore.

We expect that these results can be even further optimized by e.g. hand selecting the filters for PROTO, by adjusting for the distribution changes in the new session, e.g. by adapting the LDA as presented in [23], or by applying advanced covariate-shift compensation methods like [24].

Future work will aim to extend the presented zero training idea towards BCI novices.

## References

[1] J. R. Wolpaw, N. Birbaumer, D. J. McFarland, G. Pfurtscheller, and T. M. Vaughan, "Brain-computer interfaces for communication and control", *Clin. Neurophysiol.*, 113: 767–791, 2002.

[2] N. Birbaumer, A. Kübler, N. Ghanayim, T. Hinterberger, J. Perelmouter, J. Kaiser, I. Iversen, B. Kotchoubey, N. Neumann, and H. Flor, "The Though translation device (TTD) for Completly Paralyzed Patients", *IEEE Trans. Rehab. Eng.*, 8(2): 190–193, 2000.

[3] G. Pfurtscheller and F. H. L. da Silva, "Event-related EEG/MEG synchronization and desynchronization: basic principles", *Clin. Neurophysiol.*, 110(11): 1842–1857, 1999.

[4] B. Blankertz, G. Curio, and K.-R. Müller, "Classifying Single Trial EEG: Towards Brain Computer Interfacing", in: T. G. Diettrich, S. Becker, and Z. Ghahramani, eds., *Advances in Neural Inf. Proc. Systems (NIPS 01)*, vol. 14, 157–164, 2002.

[5] L. Trejo, K. Wheeler, C. Jorgensen, R. Rosipal, S. Clanton, B. Matthews, A. Hibbs, R. Matthews, and M. Krupka, "Multimodal Neuroelectric Interface Development", *IEEE Trans. Neural Sys. Rehab. Eng.*, (11): 199–204, 2003.

[6] L. Parra, C. Alvino, A. C. Tang, B. A. Pearlmutter, N. Yeung, A. Osman, and P. Sajda, "Linear spatial integration for single trial detection in encephalography", *NeuroImage*, 7(1): 223–230, 2002.

[7] W. D. Penny, S. J. Roberts, E. A. Curran, and M. J. Stokes, "EEG-Based Communication: A Pattern Recognition Approach", *IEEE Trans. Rehab. Eng.*, 8(2): 214–215, 2000.

[8] B. Blankertz, G. Dornhege, M. Krauledat, K.-R. Müller, V. Kunzmann, F. Losch, and G. Curio, "The Berlin Brain-Computer Interface: EEG-based communication without subject training", *IEEE Trans. Neural Sys. Rehab. Eng.*, 14(2), 2006, in press.

[9] G. Pfurtscheller, C. Neuper, C. Guger, W. Harkam, R. Ramoser, A. Schlögl, B. Obermaier, and M. Pregenzer, "Current Trends in Graz Brain-computer Interface (BCI)", *IEEE Trans. Rehab. Eng.*, 8(2): 216–219, 2000.

[10] M. Schröder, T. N. Lal, T. Hinterberger, M. Bogdan, N. J. Hill, N. Birbaumer, W. Rosenstiel, and B. Schölkopf, "Robust EEG Channel Selection Across Subjects for Brain Computer Interfaces", *EURASIP Journal on Applied Signal Processing, Special Issue: Trends in Brain Computer Interfaces*, 19: 3103–3112, 2005.

[11] H. Ramoser, J. Müller-Gerking, and G. Pfurtscheller, "Optimal spatial filtering of single trial EEG during imagined hand movement", *IEEE Trans. Rehab. Eng.*, 8(4): 441–446, 2000.

[12] F. C. Meinecke, S. Harmeling, and K.-R. Müller, "Robust ICA for Super-Gaussian Sources", in: C. G. Puntonet and A. Prieto, eds., *Proc. Int. Workshop on Independent Component Analysis and Blind Signal Separation (ICA2004)*, 2004.

[13] F. C. Meinecke, S. Harmeling, and K.-R. Müller, "Inlier-based ICA with an application to super-imposed images", *Int. J. of Imaging Systems and Technology*, 2005.

[14] G. Dornhege, B. Blankertz, G. Curio, and K.-R. Müller, "Boosting bit rates in non-invasive EEG single-trial classifications by feature combination and multi-class paradigms", *IEEE Trans. Biomed. Eng.*, 51(6): 993–1002, 2004.

[15] Z. J. Koles and A. C. K. Soong, "EEG source localization: implementing the spatio-temporal decomposition approach", *Electroencephalogr. Clin. Neurophysiol.*, 107: 343–352, 1998.

[16] G. Dornhege, B. Blankertz, M. Krauledat, F. Losch, G. Curio, and K.-R. Müller, "Combined optimization of spatial and temporal filters for improving Brain-Computer Interfacing", *IEEE Trans. Biomed. Eng.*, 2006, accepted.

[17] S. Lemm, B. Blankertz, G. Curio, and K.-R. Müller, "Spatio-Spectral Filters for Improved Classification of Single Trial EEG", *IEEE Trans. Biomed. Eng.*, 52(9): 1541–1548, 2005.

[18] R. Tomioka, G. Dornhege, G. Nolte, K. Aihara, and K.-R. Müller, "Optimizing Spectral Filter for Single Trial EEG Classification", in: *Lecture Notes in Computer Science*, Springer-Verlag Heidelberg, 2006, to be presented at 28th Annual Symposium of the German Association for Pattern Recognition (DAGM 2006).

[19] R. O. Duda, P. E. Hart, and D. G. Stork, *Pattern Classification*, Wiley & Sons, 2nd edn., 2001.

[20] T. Cox and M. Cox, *Multidimensional Scaling*, Chapman & Hall, London, 2001.

[21] S. Harmeling, G. Dornhege, D. Tax, F. C. Meinecke, and K.-R. Müller, "From outliers to prototypes: ordering data", *Neurocomputing*, 2006, in press.

[22] G. Dornhege, B. Blankertz, G. Curio, and K.-R. Müller, "Combining Features for BCI", in: S. Becker, S. Thrun, and K. Obermayer, eds., *Advances in Neural Inf. Proc. Systems (NIPS 02)*, vol. 15, 1115–1122, 2003.

[23] P. Shenoy, M. Krauledat, B. Blankertz, R. P. N. Rao, and K.-R. Müller, "Towards Adaptive Classification for BCI", *J. Neural Eng.*, 3: R13–R23, 2006.

[24] S. Sugiyama and K.-R. Müller, "Input-Dependent Estimation of Generalization Error under Covariate Shift", *Statistics and Decisions*, 2006, to appear.
